# Algorithms for Independent Components Analysis and Higher Order Statistics

**Daniel D. Lee**
Bell Laboratories
Lucent Technologies
Murray Hill, NJ 07974

**Uri Rokni and Haim Sompolinsky**
Racah Institute of Physics and
Center for Neural Computation
Hebrew University
Jerusalem, 91904, Israel

## Abstract

A latent variable generative model with finite noise is used to describe several different algorithms for Independent Components Analysis (ICA). In particular, the Fixed Point ICA algorithm is shown to be equivalent to the Expectation-Maximization algorithm for maximum likelihood under certain constraints, allowing the conditions for global convergence to be elucidated. The algorithms can also be explained by their generic behavior near a singular point where the size of the optimal generative bases vanishes. An expansion of the likelihood about this singular point indicates the role of higher order correlations in determining the features discovered by ICA. The application and convergence of these algorithms are demonstrated on a simple illustrative example.

## Introduction

Independent Components Analysis (ICA) has generated much recent theoretical and practical interest because of its successes on a number of different signal processing problems. ICA attempts to decompose the observed data into components that are as statistically independent from each other as possible, and can be viewed as a nonlinear generalization of Principal Components Analysis (PCA). Some applications of ICA include blind separation of audio signals, beamforming of radio sources, and discovery of features in biomedical traces [1].

There have also been a number of approaches to deriving algorithms for ICA [2, 3, 4]. Fundamentally, they all consider the problem of recovering independent source signals $\{\vec{s}\}$ from observations $\{\vec{x}\}$ such that:

$$x_i = \sum_{j=1}^{M} W_{ij} s_j, \ i = 1..N \tag{1}$$

Here, $W_{ij}$ is a $N \times M$ mixing matrix where the number of sources $M$ is not greater than the dimensionality $N$ of the observations. Thus, the columns of $W$ represent the different independent features present in the observed data.

Bell and Sejnowski formulated their Infomax algorithm for ICA as maximizing the mutual information between the data and a nonlinearly transformed version of the data [5]. The

covariant version of this algorithm uses the natural gradient of the mutual information to iteratively update the estimate for the demixing matrix $W^{-1}$ in terms of the estimated components $s = W^{-1}x$ [6]:

$$\Delta W^{-1} \propto \left[ I - \langle g(s)s^T \rangle \right] W^{-1}, \tag{2}$$

The nonlinearity $g(s)$ differentiates the features learned by the Infomax ICA algorithm from those found by conventional PCA. Fortunately, the exact form of the nonlinearity used in Eq. 2 is not crucial for the success of the algorithm, as long as it preserves the sub-Gaussian or super-Gaussian nature of the sources [7].

Another approach to ICA due to Hyvarinen and Oja was derived from maximizing objective functions motivated by projection pursuit [8]. Their Fixed Point ICA algorithm attempts to self-consistently solve for the extremum of a nonlinear objective function. The simplest formulation considers a single source $M = 1$ so that the mixing matrix is a single vector $w$, constrained to be unit length $|w| = 1$. Assuming the data is first preprocessed and whitened, the Fixed Point ICA algorithm iteratively updates the estimate of $w$ as follows:

$$
\begin{aligned}
w &\leftarrow \langle xg(w^T x) \rangle - \lambda_G w \\
w &\leftarrow \frac{w}{|w|},
\end{aligned}
\tag{3}
$$

where $g(w^T x)$ is a nonlinear function and $\lambda_G$ is a constant given by the integral over the Gaussian:

$$\lambda_G = \frac{1}{\sqrt{2\pi}} \int_{-\infty}^{\infty} d\eta \, e^{-\eta^2/2} \, g'(\eta). \tag{4}$$

The Fixed Point algorithm can be extended to an arbitrary number $M \leq N$ of sources by using Eq. 3 in a serial deflation scheme. Alternatively, the $M$ columns of the mixing matrix $W$ can be updated simultaneously by orthogonalizing the $N \times M$ matrix:

$$W \leftarrow \langle xg(W^T x)^T \rangle - \lambda_G W. \tag{5}$$

Under the assumption that the observed data match the underlying ICA model, $x = Ws$, it has been shown that the Fixed Point algorithm converges locally to the correct solution with at least quadratic convergence. However, the global convergence of the generic Fixed Point ICA algorithm is uncertain. This is in contrast to the gradient-based Infomax algorithm whose convergence is guaranteed as long as a sufficiently small step size is chosen.

In this paper, we first review the latent variable generative model framework for Independent Components Analysis. We then consider the generative model in the presence of finite noise, and show how the Fixed Point ICA algorithm can be related to an Expectation-Maximization algorithm for maximum likelihood. This allows us to elucidate the conditions under which the Fixed Point algorithm is guaranteed to globally converge. Assuming that the data are indeed generated from independent components, we derive the optimal parameters for convergence. We also investigate how the optimal size of the ICA mixing matrix varies as a function of the added noise, and demonstrate the presence of a singular point. By expanding the likelihood about this singular point, the behavior of the ICA algorithms can be related to the higher order statistics present in the data. Finally, we illustrate the application and convergence of these ICA algorithms on some artificial data.

## Generative model

A convenient method for interpreting the different ICA algorithms is in terms of the hidden, or latent, variable generative model shown in Fig. 1 [9, 10]. The hidden variables $\{s_j\}$

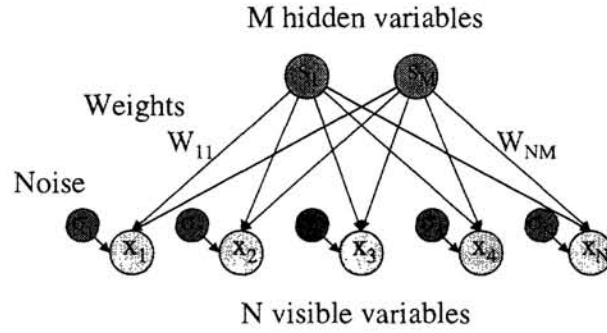

N visible variables

Figure 1: Generative model for ICA algorithms. $s$ are the hidden variables, $\sigma$ are additive Gaussian noise terms, and $x = Ws + \sigma$ are the visible variables.

correspond to the different independent components and are assumed to have the factorized non-Gaussian prior probability distribution:

$$P(s) = \prod_{j=1}^{M} e^{-F(s_j)}. \tag{6}$$

Once the hidden variables are instantiated, the visible variables $\{x_i\}$ are generated via a linear mapping through the generative weights $W$:

$$P(x|s) = \prod_{i=1}^{N} \frac{1}{\sqrt{2\pi\sigma^2}} \exp\left[-\frac{1}{2\sigma^2}(x_i - \sum_j W_{ij}s_j)^2\right], \tag{7}$$

where $\sigma^2$ is the variance of the Gaussian noise added to the visible variables.

The probability of the data given this model is then calculated by integrating over all possible values of the hidden variables:

$$P(x) = \int ds \, P(s)P(x|s) = \frac{1}{(2\pi\sigma^2)^{N/2}} \int ds \, \exp\left[-F(s) - \frac{1}{2\sigma^2}(x - Ws)^2\right] \tag{8}$$

In the limit that the added noise vanishes, $\sigma^2 \rightarrow 0$, it has previously been shown that maximizing the likelihood of Eq. 8 is equivalent to the Infomax algorithm in Eq. 2 [11]. In the following analysis, we will consider the situation when the variance of the noise is nonzero, $\sigma^2 \neq 0$.

## Expectation-Maximization

We assume that the data has initially been preprocessed and spherized: $\langle x_i x_j \rangle = \delta_{ij}$. Unfortunately, for finite noise $\sigma^2$ and an arbitrary prior $F(s_j)$, deriving a learning rule for $W$ in closed form is analytically intractable. However, it becomes possible to derive a simple Expectation–Maximization (EM) learning rule under the constraint:

$$W = \xi W_0, \quad W_0^T W_0 = I, \tag{9}$$

which implies that $W$ is orthogonal, and $\xi$ is the length of the individual columns of $W$. Indeed, for data that obeys the ICA model, $x = Ws$, it can be shown that the optimal $W$ must satisfy this orthogonality condition. By assuming the constraint in Eq. 9 for arbitrary data, the posterior distribution $P(s|x)$ becomes conveniently factorized:

$$P(s|x) \propto \prod_{j=1}^{M} \exp\left[-F(s_j) + \frac{1}{\sigma^2}[(W^T x)_j s_j - \frac{1}{2}\xi^2 s_j^2]\right]. \tag{10}$$

For the E-step, this factorized form allows the expectation function $\int ds\, P(s|x)s = g(W^T x)$ to be analytically evaluated. This expectation is then used in the M-step to find the new estimate $W'$:

$$\langle xg(W^T x)^T \rangle - \Lambda_S W' = 0, \tag{11}$$

where $\Lambda_S$ is a symmetric matrix of Lagrange multipliers that constrain the new $W'$ to be orthogonal. Eq. 11 is easily solved by taking the reduced singular value decomposition of the rectangular matrix:

$$UDV^T = \langle xg(W^T x)^T \rangle, \tag{12}$$

where $U^T U = VV^T = I$ and $D$ is a diagonal $M \times M$ matrix. Then the solution for the EM estimate of the mixing matrix is given by:

$$W' = \xi UV^T \tag{13}$$

$$\Lambda_S = \frac{1}{\xi} UDU^T. \tag{14}$$

As a specific example, consider the following prior for binary hidden variables: $P(s) = \frac{1}{2}[\delta(s-1) + \delta(s+1)]$. In this case, the expectation $\int ds\, P(s|x)s = \tanh(W^T x/\sigma^2)$ and so the EM update rule is given by orthogonalizing the matrix:

$$W \leftarrow \left\langle x \tanh(\frac{1}{\sigma^2} W^T x) \right\rangle. \tag{15}$$

**Fixed Point ICA**

Besides the presence of the linear term $\lambda_G W$ in Eq. 5, the EM update rule looks very much like that of the Fixed Point ICA algorithm. It turns out that without this linear term, the convergence of the naive EM algorithm is much slower than that of Eq. 5. Here we show that it is possible to interpret the role of this linear term in the Fixed Point ICA algorithm within the framework of this generative model.

Suppose that the distribution of the observed data $P_D(x)$ is actually a mixture between an isotropic distribution $P_0(x)$ and a non-isotropic distribution $P_1(x)$:

$$P_D(x) = \alpha P_0(x) + (1 - \alpha)P_1(x). \tag{16}$$

Because the isotropic part does not break rotational symmetry, it does not affect the choice of the directions of the learned basis $W$. Thus, it is more efficient to apply the learning algorithm to only the non-isotropic portion of the distribution, $P_1(x) \propto P_D(x) - \alpha P_0(x)$, rather than to the whole observed distribution $P_D(x)$. Applying EM to $P_1(x)$ results in a correction term arising from the subtracted isotropic distribution. With this correction, the EM update becomes:

$$W \leftarrow \langle xg(W^T x) \rangle - \alpha \lambda_G W \tag{17}$$

which is equivalent to the Fixed Point ICA algorithm when $\alpha = 1$.

Unfortunately, it is not clear how to compute an appropriate value for $\alpha$ to use in fitting data. Taking a very small value, $\alpha \ll 1$, will result in a learning rule that is very similar to the naive EM update rule. This implies that the algorithm will be guaranteed to monotonically converge, albeit very slowly, to a local maximum of the likelihood. On the other hand, choosing a large value, $\alpha \gg 1$, will result in a subtracted probability density $P_1(x)$ that is negative everywhere. In this case, the algorithm will converge slowly to a local minimum of the likelihood. For the Fixed Point algorithm which operates in the intermediate regime, $\alpha \approx 1$, the algorithm is likely to converge most rapidly. However, it is also in this situation that the subtracted density $P_1(x)$ could have both positive and negative regions, and the algorithm is no longer guaranteed to converge.

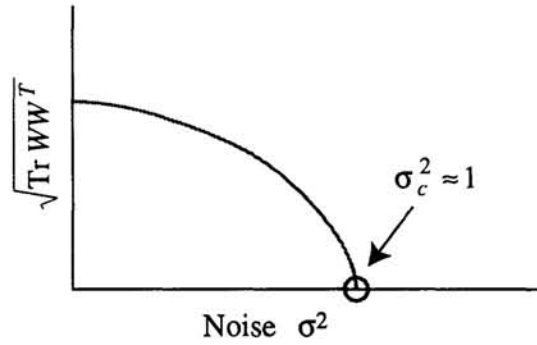

Figure 2: Size of the optimal generative bases as a function of the added noise $\sigma^2$, showing the singular point behavior around $\sigma_c^2 \approx 1$.

**Optimal value of $\alpha$**

In order to determine the optimal value of $\alpha$, we make the assumption that the observed data obeys the ICA model, $x = As$. Note that the statistics of the sources in the data need not match the assumed prior distribution of the sources in the generative model Eq. 6. With this assumption, which is not related to the mixture assumption in Eq. 16, it is easy to show that $W = A$ is a fixed point of the algorithm. By analyzing the behavior of the algorithm in the vicinity of this fixed point, a simple expression emerges for the change in deviations from this fixed point, $\delta W$, after a single iteration of Eq. 17:

$$\delta W_{ij} \leftarrow \frac{\langle g'(s)\rangle - \alpha\lambda_G}{\langle sg(s)\rangle - \alpha\lambda_G}\delta W_{ij} + O(\delta W^3) \tag{18}$$

where the averaging here is over the true source distribution, assumed for simplicity to be identical for all sources. Thus, the algorithm converges most rapidly if one chooses:

$$\alpha_{opt} = \frac{\langle g'(s)\rangle}{\lambda_G}, \tag{19}$$

so that the local convergence is cubic. From Eq. 18 one can show that the condition for the stability of the fixed point is given by $\alpha < \alpha_c$, where:

$$\alpha_c = \frac{\langle sg(s) + g'(s)\rangle}{2\lambda_G}. \tag{20}$$

Thus, for $\alpha = 0$, the stability criterion in Eq. 18 is equivalent to $\langle sg(s)\rangle > \langle g'(s)\rangle$. For the cubic nonlinearity $g(s) = s^3$, this implies that the algorithm will find the true independent features only if the source distribution has positive kurtosis.

**Singular point expansion**

Let us now consider how the optimal size $\xi$ of the weights $W$ varies as a function of the noise parameter $\sigma^2$. For very small $\sigma^2 \ll 1$, the weights $W$ are approximately described by the Infomax algorithm of Eq. 2, and the lengths of the columns should be unity in order to match the covariance of the data. For large $\sigma^2 \gg 1$, however, the optimal size of the weights should be very small because the covariance of the noise is already larger than that of the data. In fact, for Factor Analysis which is a special case of the generative model with $F(s) = \frac{1}{2}s^2$ in Eq. 6, it can be shown that the weights are exactly zero, $W = 0$, for $\sigma^2 > 1$.

Thus, the size of the optimal generative weights $W$ varies with $\sigma^2$ as shown qualitatively in Fig. 2. Above a certain critical noise value $\sigma_c^2 \approx 1$, the weights are exactly equal to

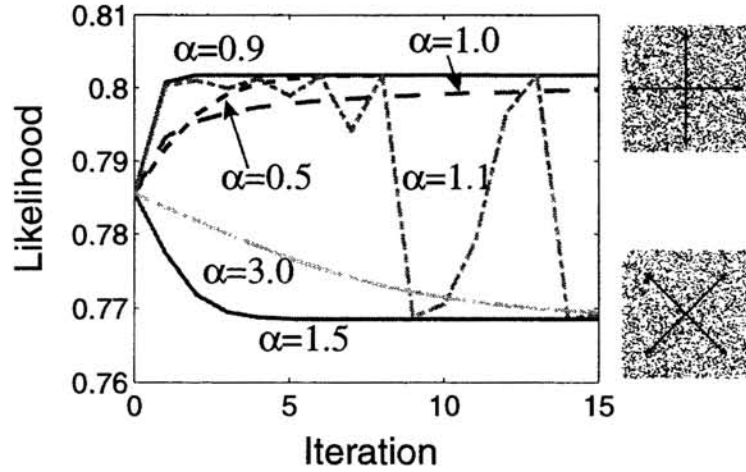

Figure 3: Convergence of the modified EM algorithm as a function of $\alpha$. With $g(s) = \tanh(s)$ as the nonlinearity, the likelihood $\langle \ln\cosh(W^T x)\rangle$ is plotted as a function of the iteration number. The optimal basis $W$ are plotted on the two-dimensional data distribution when the likelihood is maximized (top) and minimized (bottom).

zero, $W = 0$. Only below this critical value do the weights become nonzero. We expand the likelihood of the generative model in the vicinity of this singular point. This expansion is well-behaved because the size of the generative weights $W$ acts as a small perturbative parameter in this expansion. The log likelihood of the model around this singular value is then given by:

$$
\begin{aligned}
L = \ & -\frac{1}{4}\mathrm{Tr}\left[WW^T - (1-\sigma^2)I\right]^2 \\
& + \frac{1}{4!}\sum_{ijklm} \mathrm{kurt}(s_m)\,\langle x_i x_j x_k x_l\rangle_c\, W_{im}W_{jm}W_{km}W_{lm} \\
& + O(1-\sigma^2)^3,
\end{aligned}
\tag{21}
$$

where $\mathrm{kurt}(s_m)$ represents the kurtosis of the prior distribution over the hidden variables. Note that this expansion is valid for any symmetric prior, and differs from other expansions that assume small deviations from a Gaussian prior [12, 13]. Eq. 21 shows the importance of the fourth-order cumulant of the observed data in breaking the rotational degeneracy of the weights $W$. The generic behavior of ICA is manifest in optimizing the cumulant term in Eq.21, and again depends crucially on the sign of the kurtosis that is used for the prior.

## Example with artificial data

As an illustration of the convergence of the algorithm in Eq. 17, we consider the simple two-dimensional uniform distribution:

$$
P(x_1, x_2) = \begin{cases} 1/12, & -\sqrt{3} \le x_1, x_2 \le \sqrt{3} \\ 0, & \text{otherwise} \end{cases}
\tag{22}
$$

With $g(s) = \tanh(s)$ as the nonlinearity, Fig. 3 shows how the overall likelihood converges for different values of the parameter $\alpha$ as the algorithm is iterated. For $\alpha \le 1.0$, the algorithm converges to a maximum of the likelihood, with the fastest convergence at $\alpha_{opt} = 0.9$. However, for $\alpha > 1.2$, the algorithm converges to a minimum of the likelihood. At an intermediate value, $\alpha = 1.1$, the likelihood does not converge at all, fluctuating wildly between the maximum and minimum likelihood solutions. The maximum

likelihood solution shows the basis vectors in $W$ aligned with the sides of the square distribution, whereas the minimum likelihood solution has the basis aligned with the diagonals. These solutions can also be understood as maximizing and minimizing the kurtosis terms in Eq. 21.

## Discussion

The utility of the latent variable generative model is demonstrated on deriving algorithms for ICA. By constraining the generative weights to be orthogonal, an EM algorithm is analytically obtained. By interpreting the data to be fitted as a mixture of isotropic and non-isotropic parts, a simple correction to the EM algorithm is derived. Under certain conditions, this modified algorithm is equivalent to the Fixed Point ICA algorithm, and converges much more rapidly than the naive EM algorithm. The optimal parameter for convergence is derived assuming the data is consistent with the ICA generative model. There also exists a critical value for the noise parameter in the generative model, about which a controlled expansion of the likelihood is possible. This expansion makes clear the role of higher order statistics in determining the generic behavior of different ICA algorithms.

We acknowledge the support of Bell Laboratories, Lucent Technologies, the US-Israel Binational Science Foundation, and the Israel Science Foundation. We also thank Hagai Attias, Simon Haykin, Juha Karhunen, Te-Won Lee, Erkki Oja, Sebastian Seung, Boris Shraiman, and Oren Shriki for helpful discussions.

## References

[1] Haykin, S (1999). *Neural networks: a comprehensive foundation.* 2nd ed., Prentice-Hall, Upper Saddle River, NJ.

[2] Jutten, C & Herault, J (1991). Blind separation of sources, part I: An adaptive algorithm based on neuromimetic architecture. *Signal Processing* **24**, 1–10.

[3] Comon, P (1994). Independent component analysis: a new concept? *Signal Processing* **36**, 287–314.

[4] Roth, Z & Baram, Y (1996). Multidimensional density shaping by sigmoids. *IEEE Trans. Neural Networks* **7**, 1291–1298.

[5] Bell, AJ & Sejnowski, TJ (1995). An information maximization approach to blind separation and blind deconvolution. *Neural Computation* **7**, 1129–1159.

[6] Amari, S, Cichocki, A & Yang, H (1996). A new learning algorithm for blind signal separation. *Advances in Neural Information Processing Systems* **8**, 757–763.

[7] Lee, TW, Girolami, M, & Sejnowski, TJ (1999). Independent component analysis using an extended infomax algorithm for mixed sub-gaussian and super-gaussian sources. *Neural Computation* **11**, 609–633.

[8] Hyvarinen, A & Oja, E (1997). A fast fixed-point algorithm for independent component analysis. *Neural Computation* **9**, 1483–1492.

[9] Hinton, G & Ghahramani, Z (1997). Generative models for discovering sparse distributed representations. *Philosophical Transactions Royal Society B* **352**, 1177–1190.

[10] Attias, H (1998). Independent factor analysis. *Neural Computation* **11**, 803–851.

[11] Pearlmutter, B & Parra, L (1996). A context-sensitive generalization of ICA. In ICONIP '96, 151–157.

[12] Nadal, JP & Parga, N (1997). Redundancy reduction and independent component analysis: conditions on cumulants and adaptive approaches. *Neural Computation* **9**, 1421–1456.

[13] Cardoso, JF (1999). High-order contrasts for independent component analysis. *Neural Computation* **11**, 157–192.